# Inferring Elapsed Time
# from Stochastic Neural Processes

**Misha B. Ahrens and Maneesh Sahani**
Gatsby Computational Neuroscience Unit, UCL
Alexandra House, 17 Queen Square, London, WC1N 3AR
{ahrens,maneesh}@gatsby.ucl.ac.uk

## Abstract

Many perceptual processes and neural computations, such as speech recognition, motor control and learning, depend on the ability to measure and mark the passage of time. However, the processes that make such temporal judgements possible are unknown. A number of different hypothetical mechanisms have been advanced, all of which depend on the known, temporally predictable evolution of a neural or psychological state, possibly through oscillations or the gradual decay of a memory trace. Alternatively, judgements of elapsed time might be based on observations of temporally structured, but *stochastic* processes. Such processes need not be specific to the sense of time; typical neural and sensory processes contain at least some statistical structure across a range of time scales. Here, we investigate the statistical properties of an estimator of elapsed time which is based on a simple family of stochastic process.

## 1 Introduction

The experience of the passage of time, as well as the timing of events and intervals, has long been of interest in psychology, and has more recently attracted attention in neuroscience as well. Timing information is crucial for the correct functioning of a large number of processes, such as accurate limb movement, speech and the perception of speech (for example, the difference between "ba" and "pa" lies only in the relative timing of voice onsets), and causal learning.

Neuroscientific evidence that points to a specialized neural substrate for timing is very sparse, particularly when compared to the divergent set of specific mechanisms which have been theorized. One of the most influential proposals, the scalar expectancy theory (SET) of timing [1], suggests that interval timing is based on the accumulation of activity from an internal oscillatory process. Other proposals have included banks of oscillators which, when fine-tuned, produce an alignment of phases at a specified point in time that can be used to generate a neuronal spike [2]; models in which timing occurs via the characteristic and monotonic decay of memory traces [3] or reverberant activity [4]; and randomly-connected deterministic networks, which, given neuronal processes of appropriate timescales, can be shown to encode elapsed time implicitly [5].

Although this multitude of theories shows that there is little consensus on the mechanisms responsible for timing, it does point out an important fact: that timing information is present in a range of different processes, from oscillations to decaying memories and the dynamics of randomly connected neural networks. All of the theories above choose one specific such process, and suggest that observers rely on that one alone to judge time. An alternative, which we explore here, is to phrase time estimation as a statistical problem, in which the elapsed time $\Delta t$ is extracted from a collection of *stochastic* processes whose statistics are known. This is loosely analogous to accounts have appeared in the psychological literature in the form of number-of-events models [6], which suggest that the number of events in an interval influence the perception of its duration. Such models have

been related to recent psychological findings the show that the nature of the stimulus being timed affects judgments of duration [7].

Here, by contrast, we consider the properties of duration estimators that are based on more general stochastic processes. The particular stochastic processes we analyze are abstract. However, they may be seen as models both for internally-generated neural processes, such as (spontaneous) network activity and local field potentials, and for sensory processes, in the form of externally-driven neural activity, or (taking a functional view) in the form of the stimuli themselves. Both neural activity and sensory input from the environment follow well-defined temporal statistical patterns, but the exploitation of these statistics has thus far not been studied as a potential substrate for timing judgements, despite being potentially attractive. Such a basis for timing is consistent with recent studies that show that the statistics of external stimuli affect timing estimates [8, 7], a behavior not captured by the existing mechanistic models. In addition, there is evidence that timing mechanisms are distributed [9] but subject to local (e.g. retinotopic or spatiotopic) biases [10]. Using the distributed time-varying processes which are already present in the brain is implementationally efficient, and lends itself straightforwardly to a distributed implementation. At the same time, it suggests a possible origin for the modality-specificity and locality of the bias effects, as different sets of processes may be exploited for different timing purposes. Here, we show primarily that interval estimates based on such processes obey a Weber-like scaling law for accuracy under a wide range of assumptions, as well as scaling with process number that is consistent with experimental observation; and we use estimation theoretic analysis to find the reasons behind the robustness of these scaling laws.

Neuronal spike trains exhibit internal dependencies on many time scales, ranging from milliseconds to tens of seconds [11, 12], so these — or, more likely, processes derived from spike trains, such as average network activity — are plausible candidates for the types of processes assumed in this paper. Likewise, sensory information too varies over a large range of temporal scales [13]. The particular stochastic processes we use here are Gaussian Processes, whose power spectra are chosen to be broad and roughly similar to those seen in natural stimuli.

## 2 The framework

To illustrate how random processes contain timing information, consider a random walk starting at the origin, and suppose that we see a snapshot of the random walk at another, unknown, point in time. If the walk were to end up very far from the origin, and if some statistics of the random walk were known, we would expect that the time difference between the two observations, $\Delta t$, must be reasonably long in comparison to the diffusion time of the process. If, however, the second point were still very close to the origin, we might assign a high probability to $\Delta t \approx 0$, but also some probability (associated with delayed return to the orgin) to $|\Delta t| > 0$. Access to more than one such random walk would lead to more accurate estimates (e.g. if two random walks had both moved very little between the two instances in time, our confidence that $\Delta t \approx 0$ would be greater). From such considerations it is evident that, on the basis of multiple stochastic processes, one can build up a probabilistic model for $\Delta t$.

To formalize these ideas, we model the random processes as a family of independent stationary Gaussian Processes (GPs, [14]). A GP is a stochastic process $y(t)$ in which any subset of observations $\{y(t), y(t'), y(t''), ...\}$ is jointly Gaussian distributed, so that the probability distribution over observations is completely specified by a mean value (here set to zero) and a covariance structure (here assumed to remain constant in time). We denote the set of processes by $\{y_i(t)\}$. Although this is not a necessity, we let each process evolve independently according to the same stochastic dynamics; thus the process values differ only due to the random effects. Mimicking the temporal statistics of natural scenes [15], we choose the dynamics to simultaneously contain multiple time scales — specifically, the power spectrum approximately follows a $1/f^2$ power law, were $f$ = frequency = $1/$(time scale). Some instances of such processes are shown in Figure 1.

Stationary Gaussian processes are fully described by the covariance function $K(\Delta t)$:

$$\langle y_i(t)y_i(t + \Delta t)\rangle = K(\Delta t)$$

so that the probability of observing a sequence of values $[y_i(t_1), y_i(t_2), ..., y_i(t_n)]$ is Gaussian distributed, with zero mean and covariance matrix $\Sigma_{n,n'} = K(t_{n'} - t_n)$.

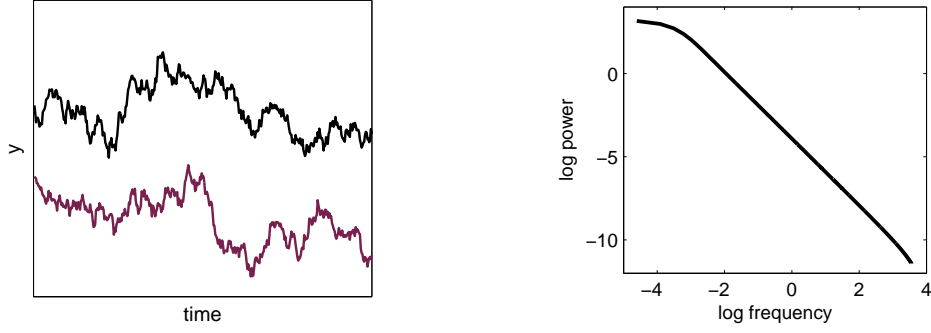

Figure 1: *Left:* Two examples of the GPs used for inference of $\Delta t$. *Right:* Their power spectrum. This is approximately a $1/f^2$ spectrum, similar to the temporal power spectrum of visual scenes.

To generate processes with multiple time scales, we approximate a $1/f^2$ spectrum with a sum over $Q$ squared exponential covariance functions:

$$K(\Delta t) = \sum_{q=1}^{Q} \alpha_q^2 \exp(-\Delta t^2/2l_q^2) + \sigma_y^2 \mathcal{I}(\Delta t)$$

Here $\sigma_y^2 \mathcal{I}(\Delta t)$ describes the instantaneous noise around the underlying covariance structure ($\mathcal{I}$ is the indicator function, which equals 1 when its argument is zero), and $l_q$ are the time scales of the component squared exponential functions. We take these to be linearly spaced, so that $l_q \propto q$. To mimic a $1/f^2$ spectrum, we choose the power of each component to be constant: $\alpha_q^2 = 1/Q$. Figure 1 shows that this choice does indeed quite accurately reproduce a $1/f^2$ power spectrum.

To illustrate how elapsed time is implicitly encoded in such stochastic processes, we infer the duration of an interval $[t, t + \Delta t]$ from two instantaneous observations of the processes, namely $\{y_i(t)\}$ and $\{y_i(t+\Delta t)\}$. For convenience, $\mathbf{y}_i$ is used to denote the vector $[y_i(t), y_i(t+\Delta t)]$. The covariance matrix $\Sigma(\Delta t)$ of $\mathbf{y}_i$, which is of size 2x2, gives rise to a likelihood of these observations,

$$P\left(\{y_i(t)\}, \{y_i(t + \Delta t)\}|\Delta t\right) \quad \propto \quad \prod_i |\Sigma|^{-1/2} \exp\left(-\frac{1}{2}\mathbf{y}_i^{\mathrm{T}}\Sigma^{-1}\mathbf{y}_i\right)$$

With the assumption of a weak prior[1], this yields a posterior distribution over $\Delta t$:

$$\Phi(\Delta t) = P(\Delta t|\{\mathbf{y}_i\}) \propto P(\Delta t) \cdot \prod_i P(\mathbf{y}_i|\Delta t)$$

$$\propto P(\Delta t) \cdot \exp\left(-\frac{1}{2}\sum_i \left[\log|\Sigma| + \mathbf{y}_i^{\mathrm{T}}\Sigma^{-1}\mathbf{y}_i\right]\right)$$

This distribution gives a probabilistic description of the time difference between two snapshots of the random processes. As we will see below (see Figure 2), this distribution tends to be centred on the true value of $\Delta t$, showing that such random processes may indeed be exploited to obtain timing information. In the following section, we explore the statistical properties of timing estimates based on $\Phi$, and show that they correspond to several experimental findings.

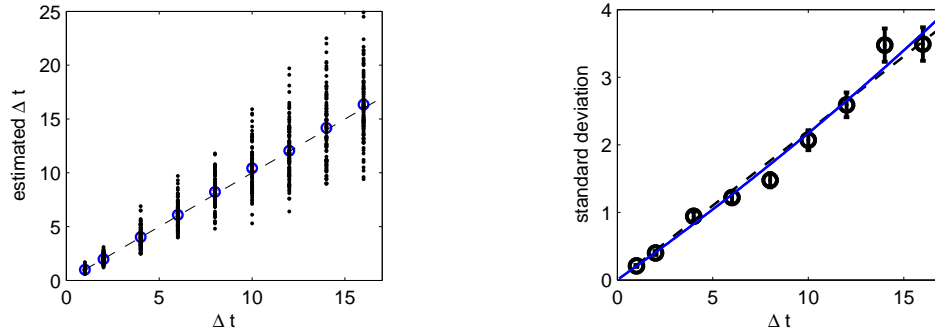

Figure 2: Statistics of the inference of $\Delta t$ from snapshots of a group of GPs. The GPs have time scales in the interval $[0.05, 50]$. *Left:* The mean estimated times (*blue*) are clustered around the true times (*dashed*). *Right:* The Weber law of timing, $\sigma \propto \Delta t$, approximately holds true for this model. The error bars are standard errors derived via a Laplace approximation to the posterior. A straight line fit is shown with a *dashed* line. The Cramer-Rao bound (*blue*), which will be derived later in the text, predicts the empirical data well.

## 3  Scaling laws and behaviour

### 3.1  Empirical demonstration of Weber's law

Many behavioral studies have shown that the standard deviation of interval estimates is proportional to the interval being judged, $\sigma \propto \Delta t$, across a wide range of timescales and tasks (e.g. [1]). Here, we show that GP-based estimates share this property under broad conditions.

To compare the behaviour of the model to experimental data, we must choose a mapping from the function $\Phi$ to a single scalar value, which will model the observer's report. A simple choice is to assume that the reported $\Delta t$ is the maximum a-posteriori (MAP) estimator based on $\Phi$, that is, $\widehat{\Delta t}_{\mathrm{MAP}} = \mathrm{argmax}_{\Delta t} \Phi(\Delta t)$. To compare the statistics of this estimator to the experimental observation, we took samples $\{y_i(t)\}$ and $\{y_i(t + \Delta t)\}$ from 50 GPs with identical $1/f^2$-like statistics containing time scales from 1 to 40 time units. 100 samples were generated for each $\Delta t$ (ranging from 1 to 16 time unis), leading to 100 estimates, $\widehat{\Delta t}_{\mathrm{MAP}}$. These estimates are plotted in Figure 2A. They are seen to follow the true $\Delta t$. Their spread around the true value increases with increasing $\Delta t$. The standard deviation of this spread is plotted in Figure 2B, and is a roughly linear function of $\Delta t$. Thus, time estimation is possible using the stochastic process framework, and the Weber law of timing holds fairly accurately.

### 3.2  Fisher Information and Weber's law

A number of questions about this Weber-like result naturally arise: Does it still hold if one changes the power spectrum of the processes? What if one changes the scale of the instantaneous noise? We increased the noise scale $\sigma_y^2$, and found that the Weber law was still approximately satisfied. When changing the power spectrum of the processes from a $1/f^2$-type spectrum to a $1/f^3$-type spectrum (by letting $\alpha_i^2 \propto l_i$ instead of $\alpha_i^2 \propto 1$), the Weber law was still approximately satisfied (Figure 3). This result may appear somewhat counter-intuitive, as one might expect that the accuracy of the estimator for $\Delta t$ would increase as the power in frequencies around $1/\Delta t$ increased; thus, changing the power spectrum to $1/f^3$ might be expected to result in more accurate estimates of large $\Delta t$ (lower frequencies) as compared to estimates of small $\Delta t$, but this was not the case.

To find reasons for this behaviour, it would useful to have an analytical expression for the relationship between the variability of the estimated duration and the true duration. This is complex, but a simpler analytical approximation to this relation can be constructed through the Cramer-Rao bound. This is a lower bound on the asymptotic variance of an unbiased Maximum Likelihood estimator of $\Delta t$ and is given by the inverse Fisher Information:

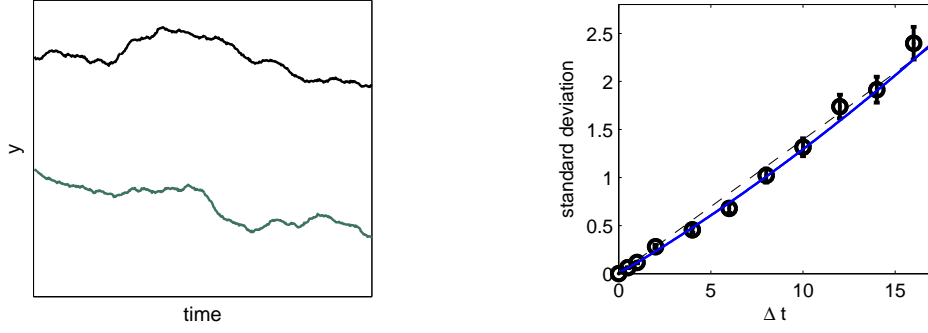

Figure 3: *Left:* Two examples of GPs with a different power spectrum ($\alpha_i^2 \propto l_i$, for $l_i \propto i$, which approximates a $1/f^3$ power spectrum, resulting in much smoother dynamics). *Right:* Inference of $\Delta t$ based on these altered processes. Note that the estimator $\widehat{\Delta t}_{\mathrm{MAP}}$ is based on the true likelihood, i.e., the new $1/f^3$ statistics. The Weber law still approximately holds, even though the dynamics is different from the initial case. The empirical standard deviation is again well predicted by the analytical Cramer-Rao bound (*blue*).

$$\mathrm{Var}(\widehat{\Delta t}) \geq 1/I_F(\Delta t)$$

The Fisher Information, assuming that the elapsed time is estimated on the basis of $N$ processes, each evolving according to covariance matrix $\Sigma(\Delta t)$, is given by the expression

$$I_F(\Delta t) = -N\left\langle \frac{\partial^2 \log P(\{\mathbf{y}_i\}|\Delta t)}{\partial \Delta t^2} \right\rangle_{\mathbf{y}} = \frac{N}{2}\mathrm{Tr}\left[\Sigma^{-1}\frac{\partial \Sigma}{\partial \Delta t}\Sigma^{-1}\frac{\partial \Sigma}{\partial \Delta t}\right] \tag{1}$$

This bound is plotted in blue in Figure 2, and again in Figure 3, and can be seen to be a good approximation to the empirical behaviour of the model.

What is the reason for the robust Weber-like behaviour? To answer this question, consider a different but related model, in which there are $N$ Gaussian processes, again labeled $i$, but each now evolving according to different covariance matrix $C_i(\Delta t)$. Previously, each process reflected structure at many timescales. In this new model, each process evolves with a single squared-exponential covariance kernel, and thus a single time-constant. This will allow us to see how each process contributes to the accuracy of the estimator.

Thus, in this model, $[C_i(\Delta t)]_{n,n'} = \alpha_i^2 \exp(-(t_{n'}-t_n)^2/2l_i^2)+\sigma_y^2\mathcal{I}(t_{n'}-t_n)$. (The power spectrum is then shaped as $\exp(-f^2 l_i^2/2)$.) The likelihood of observing the processes at two instances is now

$$P\left(\{y_i(t)\}, \{y_i(t+\Delta t)\}|\Delta t\right) \quad \propto \quad \prod_i |C_i|^{-1/2}\exp\left(-\frac{1}{2}\mathbf{y}_i^{\mathrm{T}}C_i^{-1}\mathbf{y}_i\right) \tag{2}$$

This model shows very similar behaviour to the original model, but is somewhat less natural. Its advantage lies in the fact that the Fisher Information can now be decomposed as a sum over different time scales,

$$I_F(\Delta t) = \sum_i I_{F,i} = \frac{1}{2}\sum_i \mathrm{Tr}\left[C_i^{-1}\frac{\partial C_i}{\partial \Delta t}C_i^{-1}\frac{\partial C_i}{\partial \Delta t}\right]$$

Using the Fisher Information to plot Cramer-Rao bounds for different types of processes $\{y_i(t)\}$ (Figure 4, dashed lines), we first note that the bounds are all relatively close to linear, even though the parameters governing the processes are very different. In particular, we tested both linear spacing of time scales ($l_i \propto i$) and quadratic spacing ($l_i \propto i^2$), and we tested a constant power distribution

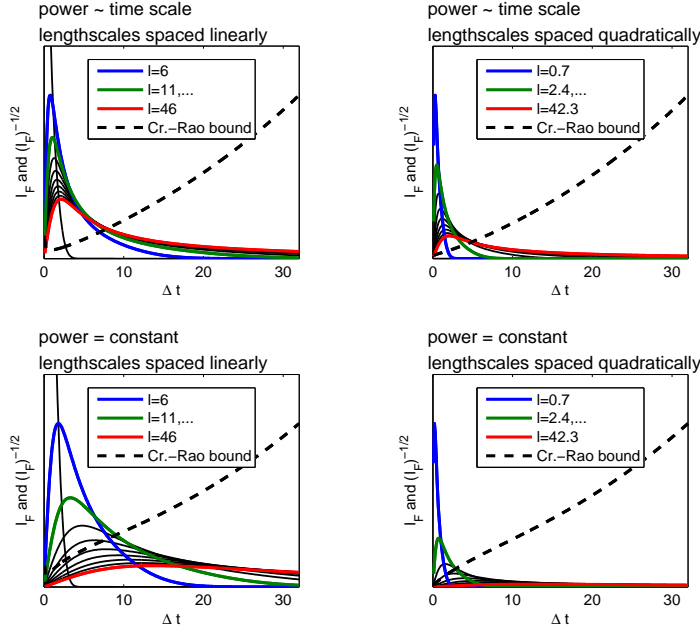

Figure 4: Fisher Information and Cramer-Rao bounds for the model of equation 2. The Cramer-Rao bound is the square root of the inverse of the sum of all the Fisher Information curves (note that only a few Fisher Information curves are shown). The noise scale $\sigma_y^2 = 0.1$, and the time scales are either $l_i = i, i = 1, 2, \ldots, 50$ (linear) or $l_i = i^2/50, i = 1, 2, \ldots, 50$ (quadratic). The power of each process is either $\alpha_i^2 = 1$ (constant) or $\alpha_i^2 = l_i$. The graphs show that each time scale contributes to the estimation of a wide range of $\Delta t$, and that the Cramer-Rao bounds are all fairly linear, leading to a robust Weber-like behaviour of the estimator of elapsed time.

($\alpha_i = 1$) and a power distribution where slower processes have more power ($\alpha_i^2 \propto l_i$). None of these manipulations caused the Cramer-Rao bound to deviate much from linearity.

Next, we can evaluate the contribution of each time scale to the accuracy of estimates of $\Delta t$, by inspecting the Fisher Information $I_{F,i}$ of a given process $\mathbf{y}_i$. Figure 4 shows that (contrary to the intuition that time scales close to $\Delta t$ contribute most to the estimation of $\Delta t$) a process evolving at a certain time scale $l_j$ contributes to the estimation of elapsed time $\Delta t$ even if $\Delta t$ is much smaller than $l_j$ (indeed, the peak of $I_{F,j}$ does not lie at $l_j$, but below it). This lies at the heart of the robust Weber-like behaviour: the details of the distribution of time scales do not matter much, because each time scale contributes to the estimation of a wide range of $\Delta t$. For similar reasons, the distribution of power does not drastically affect the Cramer-Rao bound. From the graphs of $I_{F,i}$, it is evident that the Weber law arises from an accumulation of high values of Fisher Information at low values of $\Delta t$.

Very small values of $\Delta t$ may be an exception, if the instantaneous noise dominates the subtle changes that the processes undergo during very short periods; for these $\Delta t$, the standard deviation may rise. This is reflected by a subtle rise in some of the Cramer-Rao bounds at very low values of $\Delta t$. However, it may be assumed that the shortest times that neural systems can evaluate are no shorter than the scale of the fastest process within the system, making these small $\Delta t$'s irrelevant.

### 3.3 Dependence of timing variability on the number of processes

Increasing the number of processes, say $N_{\text{processes}}$, will add more terms to the likelihood and make the estimated $\Delta t$ more accurate. The Fisher Information (equation 1) scales with $N_{\text{processes}}$, which suggests that the standard deviation of $\widehat{\Delta t}_{\text{MAP}}$ is proportional to $1/\sqrt{N_{\text{processes}}}$; this was confirmed empirically (data not shown).

Psychologically and neurally, increasing the number of processes would correspond to adding more perceptual processes, or expanding the size of the network that is being monitored for timing estimation. Although experimental data on this issue is sparse, in [9], it is shown that unimanual rhythm tapping results in a higher variability of tapping times than bimanual rhythm tapping, and that tapping with two hands and a foot results in even lower variability.

This correlates well with the theoretical scaling behaviour of the estimator $\widehat{\Delta t}_{\mathrm{MAP}}$. Note that a similar scaling law is obtained from the Multiple Timer Model [16]. This is not a model for timing itself, but for the combination of timing estimates of multiple timers; the Multiple Timer Model combines these estimates by averaging, which is the ML estimate arising from independent draws of equal variance Gaussian random variables, also resulting in a $1/\sqrt{N}$ scaling law.

Experimentally, a slower decrease in variability than a $1/\sqrt{N}$ law was observed. This can be accounted for by assuming that the processes governing the right and left hands are dependent, so that the number of effectively independent processes grows more slowly than the number of effectors.

## 4   Conclusion

We have shown that timing information is present in random processes, and can be extracted probabilistically if certain statistics of the processes are known. A neural implementation of such a framework of time estimation could use both internally generated population activity as well as external stimuli to drive its processes.

The timing estimators considered were based on the full probability distribution of the process values at times $t$ and $t'$, but simpler estimators could also be constructed. There are two reasons for considering simpler estimators: First, simpler estimators might be more easily implemented in neural systems. Second, to calculate $\Phi(\Delta t)$, one needs all of $\{y_i(t), y_i(t')\}$, so that (at least) $\{y_i(t)\}$ has to be stored in memory. One way to construct a simpler estimator might be to select a particular class (say, a linear function of $\{\mathbf{y}_i\}$) and optimize over its parameters. Alternatively, an estimator may be based on the posterior distribution over $\Delta t$ conditioned on a reduced set of parameters, with the neglected parameters integrated out. Another route might be to consider different stochatic processes, which have more compact sufficient statistics (e.g. Brownian motion, being translationally invariant, would require only $\{y_i(t') - y_i(t)\}$ instead of $\{y_i(t), y_i(t')\}$; we have not considered such processes because they are unbounded and therefore hard to associate with sensory or neural processes). We have not addressed how a memory mechanism might be combined with the stochastic process framework; this will be explored in the future.

The intention of this paper is not to offer a complete theory of neural and psychological timing, but to examine the statistical properties of a hitherto neglected substrate for timing — stochastic processes that take place in the brain or in the sensory world. It was demonstrated that estimators based on such processes replicate several important behaviors of humans and animals. Full models might be based on the same substrate, thereby naturally incorporating the same behaviors, but contain more completely specified relations to external input, memory mechanisms, adaptive mechanisms, neural implementation, and importantly, (supervised) learning of the estimator.

The neural and sensory processes that we assume to form the basis of time estimation are, of course, not fully random. But when the deterministic structure behind a process is unknown, they can still be treated as stochastic under certain statistical rules, and thus lead to a valid timing estimator. Would the GP likelihood still apply to real neural processes or would the correct likelihood be completely different? This is unknown; however, the Multivariate Central Limit Theorem implies that sums of i.i.d. stochastic processes tend to Gaussian Processes — so that, when e.g. monitoring average neuronal activity, the correct estimator may well be based on a GP likelihood.

An issue that deserves consideration is the mixing of internal (neural) and external (sensory) processes. Since timing information is present in both sensory processes (such as sound and movement of the natural world, and the motion of one's body) and internal processes (such as fluctuations in network activity), and because stimulus statistics influence timing estimates, we propose that psychological and neural timing may make use of both types of processes. However, fluctuations in the external world do not always translate into neural fluctuations (e.g. there is evidence for a spatial

code for temporal frequency in V2 [17]), so that neural and stimulus fluctuations cannot always be treated on the same footing. We will address this issue in the future.

The framework presented here has some similarities with the very interesting and more explicitly physiological model proposed by Buonomano and colleagues [5, 18], in which time is implicitly encoded in deterministic[2] neural networks through slow neuronal time constants. However, temporal information in the network model is lost when there are stimulus-independent fluctuations in the network activity, and the network can only be used as a reliable timer when it starts from a fixed resting state, and if the stimulus is identical on every trial. The difference in our scheme is that here timing estimates are based on statistics, rather than deterministic structure, so that it is fundamentally robust to noise, internal fluctuations, and stimulus changes. The stochastic process framework is, however, more abstract and farther removed from physiology, and a neural implementation may well share some features of the network model of timing.

**Acknowledgements:** We thank Jeff Beck for useful suggestions, and Peter Dayan and Carlos Brody for interesting discussions.

## Footnotes

[1] such as $P(\Delta t) = \beta \exp(-\beta \Delta t)\Theta(\Delta t)$ with $\beta \ll 1$ and $\Theta$ the Heaviside function, or $P(\Delta t) = \mathcal{U}[0, t_{\max}]$; the details of the weak prior do not affect the results.

[2]While this model and some other previous models might also contain neuronal noise, it is the deterministic (and known) element of their behaviour which encodes time.

# References

[1] J Gibbon. Scalar expectancy theory and Weber's law in animal timing. *Psychol Rev*, 84:279–325, 1977.

[2] R C Miall. The storage of time intervals using oscillating neurons. *Neural Comp*, 1:359–371, 1989.

[3] J E R Staddon and J J Higa. Time and memory: towards a pacemaker-free theory of interval timing. *J Exp Anal Behav*, 71:215–251, 1999.

[4] G Bugmann. Towards a neural model of timing. *Biosystems*, 48:11–19, 1998.

[5] D V Buonomano and M M Merzenich. Temporal information transformed into a spatial code by a neural network with realistic properties. *Science*, 267:1028–1030, 1995.

[6] D Poynter. Judging the duration of time intervals: A process of remembering segments of experience. In I Levin and D Zakay, editors, *Time and human cognition: A life-span perspective*, pages 305–331. Elsevier, 1989.

[7] R Kanai, C L E Paffen, H Hogendoorn, and F A J Verstraten. Time dilation in dynamic visual display. *J Vision*, 6:1421–1430, 2006.

[8] D M Eagleman, P U Tse, D V Buonomano, P Janssen, A C Nobre, and A O Holcombe. Time and the brain: How subjective time relates to neural time. *J Neurosci*, pages 10369–10371, 2005.

[9] R B Ivry, T C Richardson, and L L Helmuth. Improved temporal stability in multi-effector movements. *J Exp Psychol*, 28:72–92, 2002.

[10] D Burr, A Tozzi, and M C Morrone. Neural mechanisms for timing visual events are spatially selective in real-world coordinates. *Nat Neurosci*, 10:423–425, 2007.

[11] M C Teich, C Heneghan, and S B Lowen. Fractal characted of the neural spike train in the visual system of the cat. *J Opt Soc Am A*, 14:529–546, 1997.

[12] L C Osborne, W Bialek, and S G Lisberger. Time course of information about motion direction in visual area MT of macaque monkeys. *J Neurosci*, 24:3210–3222, 2004.

[13] H Attias and C E Schreiner. Temporal low-order statistics of natural sounds. In *Advances in Neural Information Processing Systems 9*, pages 27–33, 1996.

[14] C E Rasmussen and C K I Williams. *Gaussian Processes for Machine Learning*. MIT Press, Cambridge, MA, 2006.

[15] D W Dong and J J Atick. Statistics of natural time-varying images. *Network: Computation in Neural Systems*, 6:345–358, 1995.

[16] R B Ivry and T C Richardson. Temporal control and coordination: the multiple timer model. *Brain and Cognition*, 48:117–132, 2002.

[17] K H Foster, J P Gaska, M Nagler, and D A Pollen. Spatial and temporal frequency selectivity of neurones in visual cortical areas v1 and v2 of the macaque monkey. *J Physiol*, 365:331–363, 1985.

[18] U R Karmarkar and D V Buonomano. Timing in the absence of clocks: encoding time in neural network states. *Neuron*, 53:427–438, 2007.

